# The Forgetron:
# A Kernel-Based Perceptron on a Fixed Budget

**Ofer Dekel    Shai Shalev-Shwartz    Yoram Singer**
School of Computer Science & Engineering
The Hebrew University, Jerusalem 91904, Israel
{oferd,shais,singer}@cs.huji.ac.il

## Abstract

The Perceptron algorithm, despite its simplicity, often performs well on online classification tasks. The Perceptron becomes especially effective when it is used in conjunction with kernels. However, a common difficulty encountered when implementing kernel-based online algorithms is the amount of memory required to store the online hypothesis, which may grow unboundedly. In this paper we present and analyze the Forgetron algorithm for kernel-based online learning on a fixed memory budget. To our knowledge, this is the first online learning algorithm which, on one hand, maintains a *strict* limit on the number of examples it stores while, on the other hand, entertains a relative mistake bound. In addition to the formal results, we also present experiments with real datasets which underscore the merits of our approach.

## 1   Introduction

The introduction of the Support Vector Machine (SVM) [8] sparked a widespread interest in kernel methods as a means of solving (binary) classification problems. Although SVM was initially stated as a batch-learning technique, it significantly influenced the development of kernel methods in the online-learning setting. Online classification algorithms that can incorporate kernels include the Perceptron [6], ROMMA [5], ALMA [3], NORMA [4], Ballseptron [7], and the Passive-Aggressive family of algorithms [1]. Each of these algorithms observes examples in a sequence of rounds, and constructs its classification function incrementally, by storing a subset of the observed examples in its internal memory. The classification function is then defined by a kernel-dependent combination of the stored examples. This set of stored examples is the online equivalent of the *support set* of SVMs, however in contrast to the support, it continually changes as learning progresses. In this paper, we call this set the *active set*, as it includes those examples that actively define the current classifier. Typically, an example is added to the active set every time the online algorithm makes a prediction mistake, or when its confidence in a prediction is inadequately low. A rapid growth of the active set can lead to significant computational difficulties. Naturally, since computing devices have bounded memory resources, there is the danger that an online algorithm would require more memory than is physically available. This problem becomes especially eminent in cases where the online algorithm is implemented as part of a specialized hardware system with a small memory, such as a mobile telephone or an au-

tonomous robot. Moreover, an excessively large active set can lead to unacceptably long running times, as the time-complexity of each online round scales linearly with the size of the active set.

Crammer, Kandola, and Singer [2] first addressed this problem by describing an online kernel-based modification of the Perceptron algorithm in which the active set does not exceed a predefined *budget*. Their algorithm removes redundant examples from the active set so as to make the best use of the limited memory resource. Weston, Bordes and Bottou [9] followed with their own online kernel machine on a budget. Both techniques work relatively well in practice, however they both lack a theoretical guarantee on their prediction accuracy. In this paper we present the Forgetron algorithm for online kernel-based classification. To the best of our knowledge, the Forgetron is the first online algorithm with a fixed memory budget which also entertains a formal worst-case mistake bound. We name our algorithm the Forgetron since its update builds on that of the Perceptron and since it gradually forgets active examples as learning progresses.

This paper is organized as follows. In Sec. 2 we begin with a more formal presentation of our problem and discuss some difficulties in proving mistake bounds for kernel-methods on a budget. In Sec. 3 we present an algorithmic framework for online prediction with a predefined budget of active examples. Then in Sec. 4 we derive a concrete algorithm within this framework and analyze its performance. Formal proofs of our claims are omitted due to the lack of space. Finally, we present an empirical evaluation of our algorithm in Sec. 5.

## 2    Problem Setting

Online learning is performed in a sequence of consecutive rounds. On round $t$ the online algorithm observes an instance $\mathbf{x}_t$, which is drawn from some predefined instance domain $\mathcal{X}$. The algorithm predicts the binary label associated with that instance and is then provided with the correct label $y_t \in \{-1, +1\}$. At this point, the algorithm may use the instance-label pair $(\mathbf{x}_t, y_t)$ to improve its prediction mechanism. The goal of the algorithm is to correctly predict as many labels as possible.

The predictions of the online algorithm are determined by a *hypothesis* which is stored in its internal memory and is updated from round to round. We denote the hypothesis used on round $t$ by $f_t$. Our focus in this paper is on margin based hypotheses, namely, $f_t$ is a function from $\mathcal{X}$ to $\mathbb{R}$ where $\text{sign}(f_t(\mathbf{x}_t))$ constitutes the actual binary prediction and $|f_t(\mathbf{x}_t)|$ is the confidence in this prediction. The term $yf(\mathbf{x})$ is called the *margin* of the prediction and is positive whenever $y$ and $\text{sign}(f(\mathbf{x}))$ agree. We can evaluate the performance of a hypothesis on a given example $(\mathbf{x}, y)$ in one of two ways. First, we can check whether the hypothesis makes a prediction mistake, namely determine whether $y = \text{sign}(f(\mathbf{x}))$ or not. Throughout this paper, we use $M$ to denote the number of prediction mistakes made by an online algorithm on a sequence of examples $(\mathbf{x}_1, y_1), \ldots, (\mathbf{x}_T, y_T)$. The second way we evaluate the predictions of a hypothesis is by using the *hinge-loss* function, defined as,

$$\ell\big(f; (\mathbf{x}, y)\big) \;=\; \left\{ \begin{array}{ll} 0 & \text{if } yf(\mathbf{x}) \geq 1 \\ 1 - yf(\mathbf{x}) & \text{otherwise} \end{array} \right. . \tag{1}$$

The hinge-loss penalizes a hypothesis for any margin less than 1. Additionally, if $y \neq \text{sign}(f(\mathbf{x}))$ then $\ell(f, (\mathbf{x}, y)) \geq 1$ and therefore the *cumulative hinge-loss* suffered over a sequence of examples upper bounds $M$. The algorithms discussed in this paper use kernel-based hypotheses that are defined with respect to a kernel operator $K : \mathcal{X} \times \mathcal{X} \to \mathbb{R}$ which adheres to Mercer's positivity conditions [8]. A kernel-based hypothesis takes the form,

$$f(\mathbf{x}) = \sum_{i=1}^{k} \alpha_i K(\mathbf{x}_i, \mathbf{x}) \;, \tag{2}$$

where $\mathbf{x}_1, \ldots, \mathbf{x}_k$ are members of $\mathcal{X}$ and $\alpha_1, \ldots, \alpha_k$ are real weights. To facilitate the derivation of our algorithms and their analysis, we associate a reproducing kernel Hilbert space (RKHS) with $K$ in the standard way common to all kernel methods. Formally, let $\mathcal{H}_K$ be the closure of the set of all hypotheses of the form given in Eq. (2). For any two functions, $f(\mathbf{x}) = \sum_{i=1}^{k} \alpha_i K(\mathbf{x}_i, \mathbf{x})$ and $g(\mathbf{x}) = \sum_{j=1}^{l} \beta_j K(\mathbf{z}_j, \mathbf{x})$, define the inner product between them to be, $\langle f, g \rangle = \sum_{i=1}^{k} \sum_{j=1}^{l} \alpha_i \beta_j K(\mathbf{x}_i, \mathbf{z}_j)$. This inner-product naturally induces a norm defined by $\|f\| = \langle f, f \rangle^{1/2}$ and a metric $\|f - g\| = (\langle f, f \rangle - 2\langle f, g \rangle + \langle g, g \rangle)^{1/2}$. These definitions play an important role in the analysis of our algorithms. Online kernel methods typically restrict themselves to hypotheses that are defined by some subset of the examples observed on previous rounds. That is, the hypothesis used on round $t$ takes the form, $f_t(\mathbf{x}) = \sum_{i \in I_t} \alpha_i K(\mathbf{x}_i, \mathbf{x})$, where $I_t$ is a subset of $\{1, \ldots, (t\text{-}1)\}$ and $\mathbf{x}_i$ is the example observed by the algorithm on round $i$. As stated above, $I_t$ is called the active set, and we say that example $\mathbf{x}_i$ is *active* on round $t$ if $i \in I_t$.

Perhaps the most well known online algorithm for binary classification is the Perceptron [6]. Stated in the form of a kernel method, the hypotheses generated by the Perceptron take the form $f_t(\mathbf{x}) = \sum_{i \in I_t} y_i K(\mathbf{x}_i, \mathbf{x})$. Namely, the weight assigned to each active example is either $+1$ or $-1$, depending on the label of that example. The Perceptron initializes $I_1$ to be the empty set, which implicitly sets $f_1$ to be the zero function. It then updates its hypothesis only on rounds where a prediction mistake is made. Concretely, on round $t$, if $f_t(\mathbf{x}_t) \neq y_t$ then the index $t$ is inserted into the active set. As a consequence, the size of the active set on round $t$ equals the number of prediction mistakes made on previous rounds. A relative mistake bound can be proven for the Perceptron algorithm. The bound holds for any sequence of instance-label pairs, and compares the number of mistakes made by the Perceptron with the cumulative hinge-loss of any fixed hypothesis $g \in \mathcal{H}_K$, even one defined with prior knowledge of the sequence.

**Theorem 1.** *Let $K$ be a Mercer kernel and let $(\mathbf{x}_1, y_1), \ldots, (\mathbf{x}_T, y_T)$ be a sequence of examples such that $K(\mathbf{x}_t, \mathbf{x}_t) \leq 1$ for all $t$. Let $g$ be an arbitrary function in $\mathcal{H}_K$ and define $\hat{\ell}_t = \ell\big(g; (\mathbf{x}_t, y_t)\big)$. Then the number of prediction mistakes made by the Perceptron on this sequence is bounded by, $M \leq \|g\|^2 + 2 \sum_{t=1}^{T} \hat{\ell}_t$.*

Although the Perceptron is guaranteed to be competitive with any fixed hypothesis $g \in \mathcal{H}_K$, the fact that its active set can grow without a bound poses a serious computational problem. In fact, this problem is common to most kernel-based online methods that do not explicitly monitor the size of $I_t$.

As discussed above, our goal is to derive and analyze an online prediction algorithm which resolves these problems by enforcing a *fixed* bound on the size of the active set. Formally, let $B$ be a positive integer, which we refer to as the *budget parameter*. We would like to devise an algorithm which enforces the constraint $|I_t| \leq B$ on every round $t$. Furthermore, we would like to prove a relative mistake bound for this algorithm, analogous to the bound stated in Thm. 1. Regretfully, this goal turns out to be impossible without making additional assumptions. We show this inherent limitation by presenting a simple counterexample which applies to any online algorithm which uses a prediction function of the form given in Eq. (2), and for which $|I_t| \leq B$ for all $t$. In this example, we show a hypothesis $g \in \mathcal{H}_K$ and an arbitrarily long sequence of examples such that the algorithm makes a prediction mistake on every single round whereas $g$ suffers no loss at all. We choose the instance space $\mathcal{X}$ to be the set of $B+1$ standard unit vectors in $\mathbb{R}^{B+1}$, that is $\mathcal{X} = \{e_i\}_{i=1}^{B+1}$ where $e_i$ is the vector with 1 in its $i$'th coordinate and zeros elsewhere. $K$ is set to be the standard inner-product in $\mathbb{R}^{B+1}$, that is $K(\mathbf{x}, \mathbf{x}') = \langle \mathbf{x}, \mathbf{x}' \rangle$. Now for every $t$, $f_t$ is a linear combination of at most $B$ vectors from $\mathcal{X}$. Since $|\mathcal{X}| = B + 1$, there exists a vector $\mathbf{x}_t \in \mathcal{X}$ which is currently not in the active set. Furthermore, $\mathbf{x}_t$ is orthogonal to all of the active vectors and therefore $f_t(\mathbf{x}_t) = 0$. Assume without loss of generality that the online algorithm we

are using predicts $y_t$ to be $-1$ when $f_t(\mathbf{x}) = 0$. If on every round we were to present the online algorithm with the example $(\mathbf{x}_t, +1)$ then the online algorithm would make a prediction mistake on every round. On the other hand, the hypothesis $\bar{g} = \sum_{i=1}^{B+1} e_i$ is a member of $H_K$ and attains a zero hinge-loss on every round. We have found a sequence of examples and a fixed hypothesis (which is indeed defined by more than $B$ vectors from $\mathcal{X}$) that attains a cumulative loss of zero on this sequence, while the number of mistakes made by the online algorithm equals the number of rounds. Clearly, a theorem along the lines of Thm. 1 cannot be proven.

One way to resolve this problem is to limit the set of hypotheses we compete with to a sub-set of $\mathcal{H}_K$, which would naturally exclude $\bar{g}$. In this paper, we limit the set of competitors to hypotheses with small norms. Formally, we wish to devise an online algorithm which is competitive with every hypothesis $g \in \mathcal{H}_K$ for which $\|g\| \leq U$, for some constant $U$. Our counterexample indicates that we cannot prove a relative mistake bound with $U$ set to at least $\sqrt{B+1}$, since that was the norm of $\bar{g}$ in our counterexample. In this paper we come close to this upper bound by proving that our algorithms can compete with any hypothesis with a norm bounded by $\frac{1}{4}\sqrt{(B+1)/\log(B+1)}$.

## 3 A Perceptron with "Shrinking" and "Removal" Steps

The Perceptron algorithm will serve as our starting point. Recall that whenever the Perceptron makes a prediction mistake, it updates its hypothesis by adding the element $t$ to $I_t$. Thus, on any given round, the size of its active set equals the number of prediction mistakes it has made so far. This implies that the Perceptron may violate the budget constraint $|I_t| \leq B$. We can solve this problem by removing an example from the active set whenever its size exceeds $B$. One simple strategy is to remove the oldest example in the active set whenever $|I_t| > B$. Let $t$ be a round on which the Perceptron makes a prediction mistake. We apply the following two step update. First, we perform the Perceptron's update by adding $t$ to $I_t$. Let $I'_t = I_t \cup \{t\}$ denote the resulting active set. If $|I'_t| \leq B$ we are done and we set $I_{t+1} = I'_t$. Otherwise, we apply a *removal* step by finding the oldest example in the active set, $r_t = \min I'_t$, and setting $I_{t+1} = I'_t \setminus \{r_t\}$. The resulting algorithm is a simple modification of the kernel Perceptron, which conforms with a fixed budget constraint. While we are unable to prove a mistake bound for this algorithm, it is nonetheless an important milestone on the path to an algorithm with a fixed budget and a formal mistake bound.

The removal of the oldest active example from $I_t$ may significantly change the hypothesis and effect its accuracy. One way to overcome this obstacle is to reduce the weight of old examples in the definition of the current hypothesis. By controlling the weight of the oldest active example, we can guarantee that the removal step will not significantly effect the accuracy of our predictions. More formally, we redefine our hypothesis to be,

$$ f_t = \sum_{i \in I_t} \sigma_{i,t} y_i K(\mathbf{x}_i, \cdot) \ , $$

where each $\sigma_{i,t}$ is a weight in $(0, 1]$. Clearly, the effect of removing $r_t$ from $I_t$ depends on the magnitude of $\sigma_{r_t,t}$.

Using the ideas discussed above, we are now ready to outline the Forgetron algorithm. The Forgetron initializes $I_1$ to be the empty set, which implicitly sets $f_1$ to be the zero function. On round $t$, if a prediction mistake occurs, a three step update is performed. The first step is the standard Perceptron update, namely, the index $t$ is inserted into the active set and the weight $\sigma_{t,t}$ is set to be 1. Let $I'_t$ denote the active set which results from this update, and let $f'_t$ denote the resulting hypothesis, $f'_t(\mathbf{x}) = f_t(\mathbf{x}) + y_t K(\mathbf{x}_t, \mathbf{x})$. The second step of the update is a *shrinking* step in which we scale $f'$ by a coefficient $\phi_t \in (0, 1]$. The value of

$\phi_t$ is intentionally left unspecified for now. Let $f_t''$ denote the resulting hypothesis, that is, $f_t'' = \phi_t f_t'$. Setting $\sigma_{i,t+1} = \phi_t \sigma_{i,t}$ for all $i \in I_t'$, we can write,

$$f_t''(\mathbf{x}) = \sum_{i \in I_t'} \sigma_{i,t+1} y_i K(\mathbf{x}_i, \mathbf{x}) \ .$$

The third and last step of the update is the removal step discussed above. That is, if the budget constraint is violated and $|I_t'| > B$ then $I_{t+1}$ is set to be $I_t' \setminus \{r_t\}$ where $r_t = \min I_t'$. Otherwise, $I_{t+1}$ simply equals $I_t'$. The recursive definition of the weight $\sigma_{i,t}$ can be unraveled to give the following explicit form, $\sigma_{i,t} = \prod_{j \in I_{t-1} \wedge j \geq i} \phi_j$. If the shrinking coefficients $\phi_t$ are sufficiently small, then the example weights $\sigma_{i,t}$ decrease rapidly with $t$, and particularly the weight of the oldest active example can be made arbitrarily small. Thus, if $\phi_t$ is small enough, then the removal step is guaranteed not to cause any significant damage. Alas, aggressively shrinking the online hypothesis with every update might itself degrade the performance of the online hypothesis and therefore $\phi_t$ should not be set too small. The delicate balance between safe removal of the oldest example and over-aggressive scaling is our main challenge. To formalize this tradeoff, we begin with the mistake bound in Thm. 1 and investigate how it is effected by the shrinking and removal steps.

We focus first on the removal step. Let $J$ denote the set of rounds on which the Forgetron makes a prediction mistake and define the function,

$$\Psi(\sigma, \phi, \mu) = (\sigma \phi)^2 + 2 \sigma \phi (1 - \phi \mu) \ .$$

Let $t \in J$ be a round on which $|I_t| = B$. On this round, example $r_t$ is removed from the active set. Let $\mu_t = y_{r_t} f_t'(\mathbf{x}_{r_t})$ be the signed margin attained by $f_t'$ on the active example being removed. Finally, we abbreviate,

$$\Psi_t = \begin{cases} \Psi(\sigma_{r_t,t}, \phi_t, \mu_t) & \text{if } t \in J \wedge |I_t| = B \\ 0 & \text{otherwise} \end{cases} \ .$$

Lemma 1 below states that removing example $r_t$ from the active set on round $t$ increases the mistake bound by $\Psi_t$. As expected, $\Psi_t$ decreases with the weight of the removed example, $\sigma_{r_t,t+1}$. In addition, it is clear from the definition of $\Psi_t$ that $\mu_t$ also plays a key role in determining whether $\mathbf{x}_{r_t}$ can be safely removed from the active set. We note in passing that [2] used a heuristic criterion similar to $\mu_t$ to dynamically choose which active example to remove on each online round.

Turning to the shrinking step, for every $t \in J$ we define,

$$\Phi_t = \begin{cases} 1 & \text{if } \|f_{t+1}\| \geq U \\ \phi_t & \text{if } \|f_t'\| \leq U \wedge \|f_{t+1}\| < U \\ \frac{\phi_t \|f_t'\|}{U} & \text{if } \|f_t'\| > U \wedge \|f_{t+1}\| < U \end{cases} \ .$$

Lemma 1 below also states that applying the shrinking step on round $t$ increases the mistake bound by $U^2 \log(1/\Phi_t)$. Note that if $\|f_{t+1}\| \geq U$ then $\Phi_t = 1$ and the shrinking step on round $t$ has no effect on our mistake bound. Intuitively, this is due to the fact that, in this case, the shrinking step does not make the norm of $f_{t+1}$ smaller than the norm of our competitor, $g$.

**Lemma 1.** *Let* $(\mathbf{x}_1, y_1), \ldots, (\mathbf{x}_T, y_T)$ *be a sequence of examples such that* $K(\mathbf{x}_t, \mathbf{x}_t) \leq 1$ *for all* $t$ *and assume that this sequence is presented to the Forgetron with a budget constraint* $B$. *Let* $g$ *be a function in* $\mathcal{H}_K$ *for which* $\|g\| \leq U$, *and define* $\hat{\ell}_t = \ell\big(g; (\mathbf{x}_t, y_t)\big)$. *Then,*

$$M \leq \left( \|g\|^2 + 2 \sum_{t=1}^{T} \hat{\ell}_t \right) + \left( \sum_{t \in J} \Psi_t + U^2 \sum_{t \in J} \log\left(1/\Phi_t\right) \right) \ .$$

The first term in the bound of Lemma 1 is identical to the mistake bound of the standard Perceptron, given in Thm. 1. The second term is the consequence of the removal and shrinking steps. If we set the shrinking coefficients in such a way that the second term is at most $\frac{M}{2}$, then the bound in Lemma 1 reduces to $M \leq \|g\|^2 + 2\sum_t \hat{\ell}_t + \frac{M}{2}$. This can be restated as $M \leq 2\|g\|^2 + 4\sum_t \hat{\ell}_t$, which is twice the bound of the Perceptron algorithm. The next lemma states sufficient conditions on $\phi_t$ under which the second term in Lemma 1 is indeed upper bounded by $\frac{M}{2}$.

**Lemma 2.** *Assume that the conditions of Lemma 1 hold and that $B \geq 83$. If the shrinking coefficients $\phi_t$ are chosen such that,*

$$\sum_{t \in J} \Psi_t \leq \frac{15}{32} M \qquad and \qquad \sum_{t \in J} \log(1/\Phi_t) \leq \frac{\log(B+1)}{2(B+1)} M \ ,$$

*then the following holds,* $\sum_{t \in J} \Psi_t + U^2 \sum_{t \in J} \log(1/\Phi_t) \leq \frac{M}{2}$ .

In the next section, we define the specific mechanism used by the Forgetron algorithm to choose the shrinking coefficients $\phi_t$. Then, we conclude our analysis by arguing that this choice satisfies the sufficient conditions stated in Lemma 2, and obtain a mistake bound as described above.

## 4   The Forgetron Algorithm

We are now ready to define the specific choice of $\phi_t$ used by the Forgetron algorithm. On each round, the Forgetron chooses $\phi_t$ to be the maximal value in $(0, 1]$ for which the damage caused by the removal step is still manageable. To clarify our construction, define $J_t = \{i \in J : i \leq t\}$ and $M_t = |J_t|$. In words, $J_t$ is the set of rounds on which the algorithm made a mistake up until round $t$, and $M_t$ is the size of this set. We can now rewrite the first condition in Lemma 2 as,

$$\sum_{t \in J_T} \Psi_t \leq \frac{15}{32} M_T \ . \tag{3}$$

Instead of the above condition, the Forgetron enforces the following stronger condition,

$$\forall i \in \{1, \ldots, T\}, \quad \sum_{t \in J_i} \Psi_t \leq \frac{15}{32} M_i \ . \tag{4}$$

This is done as follows. Define, $Q_i = \sum_{t \in J_{i-1}} \Psi_t$. Let $i$ denote a round on which the algorithm makes a prediction mistake and on which an example must be removed from the active set. The $i$'th constraint in Eq. (4) can be rewritten as $\Psi_i + Q_i \leq \frac{15}{32} M_i$. The Forgetron sets $\phi_i$ to be the maximal value in $(0, 1]$ for which this constraint holds, namely, $\phi_i = \max\{\phi \in (0, 1] : \Psi(\sigma_{r_i,i}, \phi, \mu_i) + Q_i \leq \frac{15}{32} M_i\}$. Note that $Q_i$ does not depend on $\phi$ and that $\Psi(\sigma_{r_i,i}, \phi, \mu_i)$ is a quadratic expression in $\phi$. Therefore, the value of $\phi_i$ can be found analytically. The pseudo-code of the Forgetron algorithm is given in Fig. 1.

Having described our algorithm, we now turn to its analysis. To prove a mistake bound it suffices to show that the two conditions stated in Lemma 2 hold. The first condition of the lemma follows immediately from the definition of $\phi_t$. Using strong induction on the size of $J$, we can show that the second condition holds as well. Using these two facts, the following theorem follows as a direct corollary of Lemma 1 and Lemma 2.

INPUT: Mercer kernel $K(\cdot,\cdot)$ ; budget parameter $B > 0$
INITIALIZE: $I_1 = \emptyset$ ; $f_1 \equiv 0$ ; $Q_1 = 0$ ; $M_0 = 0$
**For** $t = 1, 2, \ldots$
    receive instance $\mathbf{x}_t$ ; predict label: $\text{sign}(f_t(\mathbf{x}_t))$
    receive correct label $y_t$
    **If** $y_t f_t(\mathbf{x}_t) > 0$
      set $I_{t+1} = I_t$, $Q_{t+1} = Q_t$, $M_t = M_{t-1}$, and $\forall i \in I_t$ set $\sigma_{i,t+1} = \sigma_{i,t}$
    **Else**
      set $M_t = M_{t-1} + 1$
      (1) set $I'_t = I_t \cup \{t\}$
      **If** $|I'_t| \leq B$
        set $I_{t+1} = I'_t$, $Q_{t+1} = Q_t$, $\sigma_{t,t} = 1$, and $\forall i \in I_{t+1}$ set $\sigma_{i,t+1} = \sigma_{i,t}$
      **Else**
        (2) define $r_t = \min I_t$
          choose $\phi_t = \max\{\phi \in (0,1] : \Psi(\sigma_{r_t,t}, \phi, \mu_t) + Q_t \leq \frac{15}{32} M_t\}$
          set $\sigma_{t,t} = 1$ and $\forall i \in I'_t$ set $\sigma_{i,t+1} = \phi_t \sigma_{i,t}$
          set $Q_{t+1} = Q_t + \Psi_t$
        (3) set $I_{t+1} = I'_t \setminus \{r_t\}$
    define $f_{t+1} = \sum_{i \in I_{t+1}} \sigma_{i,t+1} y_i K(\mathbf{x}_i, \cdot)$

Figure 1: The Forgetron algorithm.

**Theorem 2.** *Let* $(\mathbf{x}_1, y_1), \ldots, (\mathbf{x}_T, y_T)$ *be a sequence of examples such that* $K(\mathbf{x}_t, \mathbf{x}_t) \leq 1$ *for all t. Assume that this sequence is presented to the Forgetron algorithm from Fig. 1 with a budget parameter* $B \geq 83$. *Let g be a function in* $\mathcal{H}_K$ *for which* $\|g\| \leq U$, *where* $U = \frac{1}{4}\sqrt{(B+1)/\log(B+1)}$, *and define* $\hat{\ell}_t = \ell(g; (\mathbf{x}_t, y_t))$. *Then, the number of prediction mistakes made by the Forgetron on this sequence is at most,*

$$M \leq 2\|g\|^2 + 4\sum_{t=1}^{T} \hat{\ell}_t$$

## 5 Experiments and Discussion

In this section we present preliminary experimental results which demonstrate the merits of the Forgetron algorithm. We compared the performance of the Forgetron with the method described in [2], which we abbreviate by CKS. When the CKS algorithm exceeds its budget, it removes the active example whose margin would be the largest after the removal. Our experiment was performed with two standard datasets: the MNIST dataset, which consists of 60,000 training examples, and the census-income (adult) dataset, with 200,000 examples. The labels of the MNIST dataset are the 10 digit classes, while the setting we consider in this paper is that of binary classification. We therefore generated binary problems by splitting the 10 labels into two sets of equal size in all possible ways, totaling $\binom{10}{5}/2 = 126$ classification problems. For each budget value, we ran the two algorithms on all 126 binary problems and averaged the results. The labels in the census-income dataset are already binary, so we ran the two algorithms on 10 different permutations of the examples and averaged the results. Both algorithms used a fifth degree non-homogeneous polynomial kernel. The results of these experiments are summarized in Fig. 2. The accuracy of the standard Perceptron (which does not depend on $B$) is marked in each plot

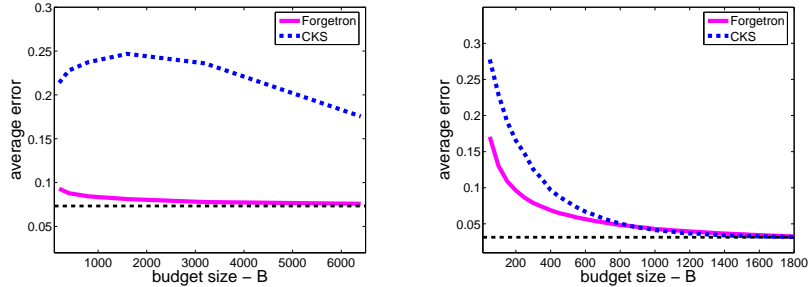

Figure 2: The error of different budget algorithms as a function of the budget size $B$ on the census-income (adult) dataset (left) and on the MNIST dataset (right). The Perceptron's active set reaches a size of 14,626 for census-income and 1,886 for MNIST. The Perceptron's error is marked with a horizontal dashed black line.

using a horizontal dashed black line. Note that the Forgetron outperforms CKS on both datasets, especially when the value of $B$ is small. In fact, on the census-income dataset, the Forgetron achieves almost the same performance as the Perceptron with only a fifth of the active examples. In contrast to the Forgetron, which performs well on both datasets, the CKS algorithm performs rather poorly on the census-income dataset. This can be partly attributed to the different level of difficulty of the two classification tasks. It turns out that the performance of CKS deteriorates as the classification task becomes more difficult. In contrast, the Forgetron seems to perform well on both easy and difficult classification tasks.

In this paper we described the Forgetron algorithm which is a kernel-based online learning algorithm with a fixed memory budget. We proved that the Forgetron is competitive with any hypothesis whose norm is upper bounded by $U = \frac{1}{4}\sqrt{(B+1)/\log(B+1)}$. We further argued that no algorithm with a budget of $B$ active examples can be competitive with every hypothesis whose norm is $\sqrt{B+1}$, on every input sequence. Bridging the small gap between $U$ and $\sqrt{B+1}$ remains an open problem. The analysis presented in this paper can be used to derive a family of online algorithms of which the Forgetron is only one special case. This family of algorithms, as well as complete proofs of our formal claims and extensive experiments, will be presented in a long version of this paper.

# References

[1] K. Crammer, O. Dekel, J. Keshet, S. Shalev-Shwartz, and Y. Singer. Online passive aggressive algorithms. Technical report, The Hebrew University, 2005.

[2] K. Crammer, J. Kandola, and Y. Singer. Online classification on a budget. *NIPS*, 2003.

[3] C. Gentile. A new approximate maximal margin classification algorithm. *JMLR*, 2001.

[4] J. Kivinen, A. J. Smola, and R. C. Williamson. Online learning with kernels. *IEEE Transactions on Signal Processing*, 52(8):2165–2176, 2002.

[5] Y. Li and P. M. Long. The relaxed online maximum margin algorithm. *NIPS*, 1999.

[6] F. Rosenblatt. The Perceptron: A probabilistic model for information storage and organization in the brain. *Psychological Review*, 65:386–407, 1958.

[7] S. Shalev-Shwartz and Y. Singer. A new perspective on an old perceptron algorithm. *COLT*, 2005.

[8] V. N. Vapnik. *Statistical Learning Theory*. Wiley, 1998.

[9] J. Weston, A. Bordes, and L. Bottou. Online (and offline) on an even tighter budget. *AISTATS*, 2005.
